# Agglomerative Information Bottleneck

Noam Slonim       Naftali Tishby*
Institute of Computer Science and
Center for Neural Computation
The Hebrew University
Jerusalem, 91904 Israel
email: {noamm,tishby}@cs.huji.ac.il

## Abstract

We introduce a novel distributional clustering algorithm that maximizes the mutual information per cluster between data and given categories. This algorithm can be considered as a bottom up hard version of the recently introduced "Information Bottleneck Method". The algorithm is compared with the top-down soft version of the information bottleneck method and a relationship between the hard and soft results is established. We demonstrate the algorithm on the *20 Newsgroups* data set. For a subset of two newsgroups we achieve compression by 3 orders of magnitudes loosing only 10% of the original mutual information.

## 1 Introduction

The problem of self-organization of the members of a set $X$ based on the similarity of the conditional distributions of the members of another set, $Y$, $\{p(y|x)\}$, was first introduced in [8] and was termed "distributional clustering".

This question was recently shown in [9] to be a special case of a much more fundamental problem: *What are the features of the variable $X$ that are relevant for the prediction of another, relevance, variable $Y$?* This general problem was shown to have a natural information theoretic formulation: *Find a compressed representation of the variable $X$, denoted $\tilde{X}$, such that the mutual information between $\tilde{X}$ and $Y$, $I(\tilde{X};Y)$, is as high as possible, under a constraint on the mutual information between $X$ and $\tilde{X}$, $I(X;\tilde{X})$.* Surprisingly, this variational problem yields an exact self-consistent equations for the conditional distributions $p(y|\tilde{x})$, $p(x|\tilde{x})$, and $p(\tilde{x})$. This constrained information optimization problem was called in [9] *The Information Bottleneck Method.*

The original approach to the solution of the resulting equations, used already in [8], was based on an analogy with the "deterministic annealing" approach to clustering (see [7]). This is a top-down hierarchical algorithm that starts from a single cluster and undergoes a cascade of cluster splits which are determined stochastically (as phase transitions) into a "soft" (fuzzy) tree of clusters.

In this paper we propose an alternative approach to the information bottleneck

problem, based on a greedy bottom-up merging. It has several advantages over the top-down method. It is fully deterministic, yielding (initially) "hard clusters", for any desired number of clusters. It gives higher mutual information per-cluster than the deterministic annealing algorithm and it can be considered as the hard (zero temperature) limit of deterministic annealing, for any prescribed number of clusters. Furthermore, using the bottleneck self-consistent equations one can "soften" the resulting hard clusters and recover the deterministic annealing solutions without the need to identify the cluster splits, which is rather tricky. The main disadvantage of this method is computational, since it starts from the limit of a cluster per each member of the set $X$.

## 1.1 The information bottleneck method

The mutual information between the random variables $X$ and $Y$ is the symmetric functional of their joint distribution,

$$I(X;Y) = \sum_{x \in X, y \in Y} p(x,y) \log \left( \frac{p(x,y)}{p(x)p(y)} \right) = \sum_{x \in X, y \in Y} p(x)p(y|x) \log \left( \frac{p(y|x)}{p(y)} \right) .$$

(1)

The objective of the information bottleneck method is to extract a compact representation of the variable $X$, denoted here by $\tilde{X}$, with minimal loss of mutual information to another, *relevance*, variable $Y$. More specifically, we want to find a (possibly stochastic) map, $p(\tilde{x}|x)$, that minimizes the (lossy) coding length of $X$ via $\tilde{X}$, $I(X; \tilde{X})$, under a constraint on the mutual information to the relevance variable $I(\tilde{X}; Y)$. In other words, we want to find an efficient representation of the variable $X$, $\tilde{X}$, such that the predictions of $Y$ from $X$ through $\tilde{X}$ will be as close as possible to the direct prediction of $Y$ from $X$.

As shown in [9], by introducing a positive Lagrange multiplier $\beta$ to enforce the mutual information constraint, the problem amounts to minimization of the Lagrangian:

$$\mathcal{L}[p(\tilde{x}|x)] = I(X; \tilde{X}) - \beta I(\tilde{X}; Y) ,$$

(2)

with respect to $p(\tilde{x}|x)$, subject to the Markov condition $\tilde{X} \to X \to Y$ and normalization.

This minimization yields directly the following self-consistent equations for the map $p(\tilde{x}|x)$, as well as for $p(y|\tilde{x})$ and $p(\tilde{x})$:

$$\begin{cases} p(\tilde{x}|x) = \frac{p(\tilde{x})}{Z(\beta,x)} \exp \left( -\beta D_{KL}[p(y|x)\|p(y|\tilde{x})] \right) \\ p(y|\tilde{x}) = \sum_x p(y|x)p(\tilde{x}|x)\frac{p(x)}{p(\tilde{x})} \\ p(\tilde{x}) = \sum_x p(\tilde{x}|x)p(x) \end{cases}$$

(3)

where $Z(\beta, x)$ is a normalization function. The functional $D_{KL}[p\|q] \equiv \sum_y p(y) \log \frac{p(y)}{q(y)}$ is the Kulback-Liebler divergence [3], which *emerges* here from the variational principle. These equations can be solved by iterations that are proved to converge for any finite value of $\beta$ (see [9]). The Lagrange multiplier $\beta$ has the natural interpretation of inverse temperature, which suggests deterministic annealing [7] to explore the hierarchy of solutions in $\tilde{X}$, an approach taken already in [8].

The variational principle, Eq.(2), determines also the shape of the annealing process, since by changing $\beta$ the mutual informations $I_X \equiv I(X; \tilde{X})$ and $I_Y \equiv I(Y; \tilde{X})$ vary such that

$$\frac{\delta I_Y}{\delta I_X} = \beta^{-1} .$$

(4)

Thus the optimal curve, which is analogous to the rate distortion function in information theory [3], follows a strictly concave curve in the $(I_X, I_Y)$ plane, called the *information plane*. Deterministic annealing, at fixed number of clusters, follows such a concave curve as well, but this curve is suboptimal beyond a certain critical value of $\beta$.

Another interpretation of the bottleneck principle comes from the relation between the mutual information and Bayes classification error. This error is bounded above and below (see [6]) by an important information theoretic measure of the class conditional distributions $p(x|y_i)$, called the *Jensen-Shannon divergence*. This measure plays an important role in our context.

The Jensen-Shannon divergence of $M$ class distributions, $p_i(x)$, each with a prior $\pi_i$, $1 \leq i \leq M$, is defined as, [6, 4].

$$JS_\pi[p_1, p_2, ..., p_M] \equiv H[\sum_{i=1}^{M} \pi_i p_i(x)] - \sum_{i=1}^{M} \pi_i H[p_i(x)] \,, \tag{5}$$

where $H[p(x)]$ is Shannon's entropy, $H[p(x)] = -\sum_x p(x) \log p(x)$. The convexity of the entropy and Jensen inequality guarantees the non-negativity of the JS-divergence.

## 1.2 The hard clustering limit

For any finite cardinality of the representation $|\tilde{X}| \equiv m$ the limit $\beta \to \infty$ of the Eqs.(3) induces a hard partition of $X$ into $m$ disjoint subsets. In this limit each member $x \in X$ belongs only to the subset $\tilde{x} \in \tilde{X}$ for which $p(y|\tilde{x})$ has the smallest $D_{KL}[p(y|x)\|p(y|\tilde{x})]$ and the probabilistic map $p(\tilde{x}|x)$ obtains the limit values 0 and 1 only.

In this paper we focus on a bottom up agglomerative algorithm for generating "good" hard partitions of $X$. We denote an m-partition of $X$, i.e. $\tilde{X}$ with cardinality $m$, also by $Z_m = \{z_1, z_2, ..., z_m\}$, in which case $p(\tilde{x}) = p(z_i)$. We say that $Z_m$ is an *optimal m-partition* (not necessarily unique) of $X$ if for every other *m-partition* of $X$, $Z'_m$, $I(Z_m; Y) \geq I(Z'_m; Y)$. Starting from the trivial $N$-partition, with $N = |X|$, we seek a sequence of merges into coarser and coarser partitions that are as close as possible to optimal.

It is easy to verify that in the $\beta \to \infty$ limit Eqs.(3) for the *m-partition distributions* are simplified as follows. Let $\tilde{x} \equiv z = \{x_1, x_2, ..., x_{|z|}\}$ , $x_i \in X$ denote a specific component (i.e. cluster) of the partition $Z_m$, then

$$\begin{cases} p(z|x) = \begin{cases} 1 & \text{if } x \in z \\ 0 & \text{otherwise} \end{cases} \forall x \in X \\ p(y|z) = \frac{1}{p(z)} \sum_{i=1}^{|z|} p(x_i, y) \ \forall y \in Y \\ p(z) = \sum_{i=1}^{|z|} p(x_i) \end{cases} \tag{6}$$

Using these distributions one can easily evaluate the mutual information between $Z_m$ and $Y$, $I(Z_m; Y)$, and between $Z_m$ and $X$, $I(Z_m; X)$, using Eq.(1).

Once any hard partition, or hard clustering, is obtained one can apply "reverse annealing" and "soften" the clusters by *decreasing* $\beta$ in the self-consistent equations, Eqs.( 3). Using this procedure we in fact recover the stochastic map, $p(\tilde{x}|x)$, from the hard partition without the need to identify the cluster splits. We demonstrate this reverse deterministic annealing procedure in the last section.

### 1.3   Relation to other work

A similar agglomerative procedure, without the information theoretic framework and analysis, was recently used in [1] for text categorization on the 20 newsgroup corpus. Another approach that stems from the distributional clustering algorithm was given in [5] for clustering dyadic data. An earlier application of mutual information for semantic clustering of words was used in [2].

## 2   The agglomerative information bottleneck algorithm

The algorithm starts with the trivial partition into $N = |X|$ clusters or components, with each component contains exactly one element of $X$. At each step we *merge* several components of the current partition into a single new component in a way that locally minimizes the loss of mutual information $I(\tilde{X}; Y) = I(Z_m; Y)$.

Let $Z_m$ be the current *m-partition* of $X$ and $Z_{\bar{m}}$ denote the new *$\bar{m}$-partition* of $X$ after the merge of several components of $Z_m$. Obviously, $\bar{m} < m$. Let $\{z_1, z_2, ..., z_k\} \subseteq Z_m$ denote the set of components to be merged, and $\bar{z}_k \in Z_{\bar{m}}$ the new component that is generated by the merge, so $\bar{m} = m - k + 1$.

To evaluate the reduction in the mutual information $I(Z_m; Y)$ due to this merge one needs the distributions that define the new *$\bar{m}$-partition*, which are determined as follows. For every $z \in Z_{\bar{m}}, z \neq \bar{z}_k$, its probability distributions $(p(z), p(y|z), p(z|x))$ remains equal to its distributions in $Z_m$. For the new component, $\bar{z}_k \in Z_{\bar{m}}$, we define,

$$\begin{cases} p(\bar{z}_k) = \sum_{i=1}^{k} p(z_i) \\ p(y|\bar{z}_k) = \frac{1}{p(\bar{z}_k)} \sum_{i=1}^{k} p(z_i, y) \quad \forall y \in Y \\ p(\bar{z}|x) = \begin{cases} 1 & \text{if } x \in z_i \text{ for some } 1 \leq i \leq k \\ 0 & \text{otherwise} \end{cases} \quad \forall x \in X \end{cases} \quad (7)$$

It is easy to verify that $Z_{\bar{m}}$ is indeed a valid *$\bar{m}$-partition* with proper probability distributions.

Using the same notations, for every *merge* we define the additional quantities:

- The **merge prior distribution**: defined by $\Pi_k \equiv (\pi_1, \pi_2, ..., \pi_k)$, where $\pi_i$ is the prior probability of $z_i$ in the merged subset, i.e. $\pi_i \equiv \frac{p(z_i)}{p(\bar{z}_k)}$.

- The **Y-information decrease**: the decrease in the mutual information $I(\tilde{X}; Y)$ due to a single merge, $\delta I_y(z_1, ..., z_k) \equiv I(Z_m; Y) - I(Z_{\bar{m}}; Y)$

- The **X-information decrease**: the decrease in the mutual information $I(\tilde{X}, X)$ due to a single merge, $\delta I_x(z_1, z_2, ..., z_k) \equiv I(Z_m; X) - I(Z_{\bar{m}}; X)$

Our algorithm is a *greedy* procedure, where in each step we perform "the best possible merge", i.e. merge the components $\{z_1, ..., z_k\}$ of the current *m-partition* which minimize $\delta I_y(z_1, ..., z_k)$. Since $\delta I_y(z_1, ..., z_k)$ can only increase with $k$ (corollary 2), for a *greedy* procedure it is enough to check only the possible merging of pairs of components of the current *m-partition*. Another advantage of merging only pairs is that in this way we go through all the possible cardinalities of $Z = \tilde{X}$, from $N$ to 1.

For a given *m-partition* $Z_m = \{z_1, z_2, ..., z_m\}$ there are $\frac{m(m-1)}{2}$ possible pairs to merge. To find "the best possible merge" one must evaluate the reduction of information $\delta I_y(z_i, z_j) = I(Z_m; Y) - I(Z_{m-1}; Y)$ for every pair in $Z_m$, which is $O(m \cdot |Y|)$ operations for every pair. However, using *proposition 1* we know that $\delta I_y(z_i, z_j) = (p(z_i) + p(z_j)) \cdot JS_{\Pi_2}(p(Y|z_i), p(Y|z_j))$, so the reduction in the mutual

information due to the merge of $z_i$ and $z_j$ can be evaluated directly (looking only at this pair) in $O(|Y|)$ operations, a reduction of a factor of $m$ in time complexity (for every merge).

---

**Input:** Empirical probability matrix $p(x,y)$, $N = |X|$, $M = |Y|$

**Output:** $Z_m$ : $m$-*partition* of $X$ into $m$ clusters, for every $1 \leq m \leq N$

**Initialization:**

- Construct $Z \equiv X$
  - For $i = 1...N$
    * $z_i = \{x_i\}$
    * $p(z_i) = p(x_i)$
    * $p(y|z_i) = p(y|x_i)$ for every $y \in Y$
    * $p(z|x_j) = 1$ if $j = i$ and 0 otherwise
  - $Z = \{z_1, ..., z_N\}$
- for every $i, j = 1...N$, $i < j$, calculate
  $d_{i,j} = (p(z_i) + p(z_j)) \cdot JS_{\Pi_2}[p(y|z_i), p(y|z_j)]$
  (every $d_{i,j}$ points to the corresponding couple in $Z$)

**Loop:**

- For $t = 1...(N-1)$
  - Find $\{\alpha, \beta\} = argmin_{i,j}\{d_{i,j}\}$
    (if there are several minima choose arbitrarily between them)
  - Merge $\{z_\alpha, z_\beta\} \Rightarrow \bar{z}$ :
    * $p(\bar{z}) = p(z_\alpha) + p(z_\beta)$
    * $p(y|\bar{z}) = \frac{1}{p(\bar{z})}(p(z_\alpha, y) + p(z_\beta, y))$ for every $y \in Y$
    * $p(\bar{z}|x) = 1$ if $x \in z_\alpha \cup z_\beta$ and 0 otherwise, for every $x \in X$
  - Update $Z = \{Z - \{z_\alpha, z_\beta\}\} \bigcup \{\bar{z}\}$
    ($Z$ is now a new $(N-t)$-*partition* of $X$ with $N-t$ clusters)
  - Update $d_{i,j}$ costs and pointers w.r.t. $\bar{z}$
    (only for couples contained $z_\alpha$ or $z_\beta$).
- **End For**

---

Figure 1: Pseudo-code of the algorithm.

## 3 Discussion

The algorithm is non-parametric, it is a simple *greedy* procedure, that depends only on the input empirical joint distribution of $X$ and $Y$. The output of the algorithm is the hierarchy of all *m-partitions* $Z_m$ of $X$ for $m = N, (N-1), ..., 2, 1$. Moreover, unlike most other clustering heuristics, it has a built in measure of efficiency even for sub-optimal solutions, namely, the mutual information $I(Z_m; Y)$ which bounds the Bayes classification error. The quality measure of the obtained $Z_m$ partition is the fraction of the mutual information between $X$ and $Y$ that $Z_m$ captures. This is given by the curve $\frac{I(Z_m;Y)}{I(X;Y)}$ vs. $m = |Z_m|$. We found that empirically this curve was *concave*. If this is always true the decrease in the mutual information at every step, given by $\delta(m) \equiv \frac{I(Z_m;Y) - I(Z_{m-1};Y)}{I(X;Y)}$ can only *increase* with decreasing $m$. Therefore, if at some point $\delta(m)$ becomes relatively high it is an indication that we have reached a value of $m$ with "meaningful" partition or clusters. Further

merging results in substantial loss of information and thus significant reduction in the performance of the clusters as features. However, since the computational cost of the final (low $m$) part of the procedure is very low we can just as well complete the merging to a single cluster.

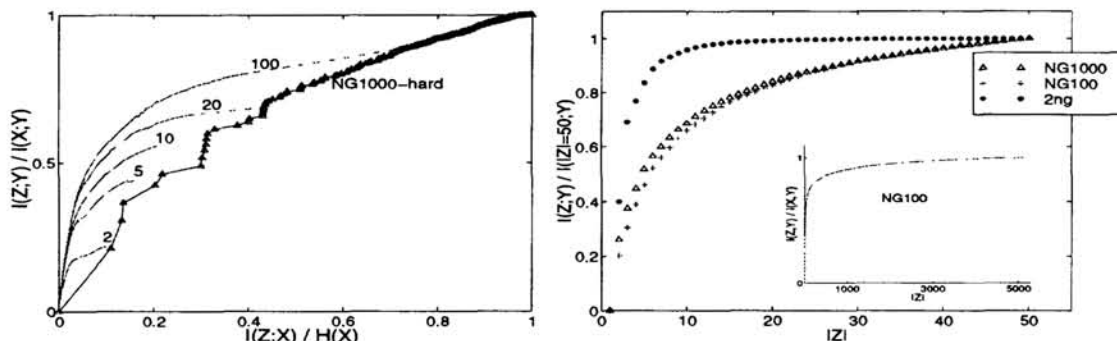

Figure 2: On the left figure the results of the agglomerative algorithm are shown in the "information plane", normalized $I(Z;Y)$ vs. normalized $I(Z;X)$ for the NG1000 dataset. It is compared to the soft version of the information bottleneck via "reverse annealing" for $|Z| = 2, 5, 10, 20, 100$ (the smooth curves on the left). For $|Z| = 20, 100$ the annealing curve is connected to the starting point by a dotted line. In this plane the hard algorithm is clearly inferior to the soft one.
On the right-hand side: $I(Z_m, Y)$ of the agglomerative algorithm is plotted vs. the cardinality of the partition $m$ for three subsets of the newsgroup dataset. To compare the performance over the different data cardinalities we normalize $I(Z_m; Y)$ by the value of $I(Z_{50}; Y)$, thus forcing all three curves to start (and end) at the same points. The predictive information on the newsgroup for NG1000 and NG100 is very similar, while for the dichotomy dataset, 2ng, a much better prediction is possible at the same $|Z|$, as can be expected for dichotomies. The inset presents the full curve of the normalized $I(Z;Y)$ vs. $|Z|$ for NG100 data for comparison. In this plane the hard partitions are superior to the soft ones.

## 4  Application

To evaluate the ideas and the algorithm we apply it to several subsets of the 20*Newsgroups* dataset, collected by Ken Lang using 20,000 articles evenly distribut­ed among 20 UseNet discussion groups (see [1]). We replaced every digit by a single character and by another to mark non-alphanumeric characters. Following this pre­processing, the first dataset contained the 530 strings that appeared more then 1000 times in the data. This dataset is referred as *NG*1000. Similarly, all the strings that appeared more then 100 times constitutes the *NG*100 dataset and it contains 5148 different strings. To evaluate also a dichotomy data we used a corpus consisting of only two discussion groups out of the 20*Newsgroups* with similar topics: *alt.atheism* and *talk.religion.misc*. Using the same pre-processing, and removing strings that occur less then 10 times, the resulting "lexicon" contained 5765 different strings. We refer to this dataset as *2ng*.

We plot the results of our algorithm on these three data sets in two different planes. First, the normalized information $\frac{I(Z;Y)}{I(X;Y)}$ vs. the size of partition of $X$ (number of clusters), $|Z|$. The greedy procedure directly tries to maximize $I(Z;Y)$ for a given $|Z|$, as can be seen by the strong concavity of these curves (figure 2, right). Indeed the procedure is able to maintain a high percentage of the relevant mutual information of the original data, while reducing the dimensionality of the "features",

$|Z|$, by several orders of magnitude.

On the right hand-side of figure 2 we present a comparison between the efficiency of the procedure for the three datasets. The two-class data, consisting of 5765 different strings, is compressed by two orders of magnitude, into 50 clusters, almost without loosing any of the mutual information about the news groups (the decrease in $I(\tilde{X};Y)$ is about 0.1%). Compression by three orders of magnitude, into 6 clusters, maintains about 90% of the original mutual information.

Similar results, even though less striking, are obtained when $Y$ contain all 20 newsgroups. The *NG100* dataset was compressed from 5148 strings to 515 clusters, keeping 86% of the mutual information, and into 50 clusters keeping about 70% of the information. About the same compression efficiency was obtained for the *NG1000* dataset.

The relationship between the soft and hard clustering is demonstrated in the *Information plane*, i.e., the normalized mutual information values, $\frac{I(Z;Y)}{I(X;Y)}$ vs. $\frac{I(Z;X)}{H(X)}$. In this plane, the soft procedure is optimal since it is a direct maximization of $I(Z;Y)$ while constraining $I(Z;X)$. While the hard partition is suboptimal in this plane, as confirmed empirically, it provides an excellent starting point for reverse annealing. In figure 2 we present the results of the agglomerative procedure for *NG1000* in the information plane, together with the reverse annealing for different values of $|Z|$. As predicted by the theory, the annealing curves merge at various critical values of $\beta$ into the globally optimal curve, which correspond to the "rate distortion function" for the information bottleneck problem. With the reverse annealing ("heating") procedure there is no need to identify the cluster splits as required in the original annealing ("cooling") procedure. As can be seen, the "phase diagram" is much better recovered by this procedure, suggesting a combination of agglomerative clustering and reverse annealing as the ultimate algorithm for this problem.

# References

[1] L. D. Baker and A. K. McCallum. Distributional Clustering of Words for Text Classification In *ACM SIGIR 98*, 1998.

[2] P. F. Brown, P.V. deSouza, R.L. Mercer, V.J. DellaPietra, and J.C. Lai. Class-based n-gram models of natural language. In *Computational Linguistics*, 18(4):467-479, 1992.

[3] T. M. Cover and J. A. Thomas. *Elements of Information Theory*. John Wiley & Sons, New York, 1991.

[4] R. El-Yaniv, S. Fine, and N. Tishby. Agnostic classification of Markovian sequences. In *Advances in Neural Information Processing (NIPS'97)* , 1998.

[5] T. Hofmann, J. Puzicha, and M. Jordan. Learning from dyadic data. In *Advances in Neural Information Processing (NIPS'98)*, 1999.

[6] J. Lin. Divergence Measures Based on the Shannon Entropy. *IEEE Transactions on Information theory*, 37(1):145–151, 1991.

[7] K. Rose. Deterministic Annealing for Clustering, Compression, Classification, Regression, and Related Optimization Problems. *Proceedings of the IEEE*, 86(11):2210–2239, 1998.

[8] F.C. Pereira, N. Tishby, and L. Lee. Distributional clustering of English words. In *30th Annual Meeting of the Association for Computational Linguistics, Columbus, Ohio*, pages 183–190, 1993.

[9] N. Tishby, W. Bialek, and F. C. Pereira. The information bottleneck method: Extracting relevant information from concurrent data. Yet unpublished manuscript, NEC Research Institute TR, 1998.